# Shrinking the Tube:
# A New Support Vector Regression Algorithm

**Bernhard Schölkopf[§,⋆], Peter Bartlett[⋆], Alex Smola[§,⋆], Robert Williamson[⋆]**
§ GMD FIRST, Rudower Chaussee 5, 12489 Berlin, Germany
⋆ FEIT/RSISE, Australian National University, Canberra 0200, Australia
bs, smola@first.gmd.de, Peter.Bartlett, Bob.Williamson@anu.edu.au

## Abstract

A new algorithm for Support Vector regression is described. For a priori chosen $\nu$, it automatically adjusts a flexible tube of minimal radius to the data such that at most a fraction $\nu$ of the data points lie outside. Moreover, it is shown how to use parametric tube shapes with non-constant radius. The algorithm is analysed theoretically and experimentally.

## 1  INTRODUCTION

Support Vector (SV) machines comprise a new class of learning algorithms, motivated by results of statistical learning theory (Vapnik, 1995). Originally developed for pattern recognition, they represent the decision boundary in terms of a typically small subset (Schölkopf et al., 1995) of all training examples, called the Support Vectors. In order for this property to carry over to the case of SV Regression, Vapnik devised the so-called $\varepsilon$-insensitive loss function $|y - f(\mathbf{x})|_\varepsilon = \max\{0, |y - f(\mathbf{x})| - \varepsilon\}$, which does not penalize errors below some $\varepsilon > 0$, chosen a priori. His algorithm, which we will henceforth call $\varepsilon$-**SVR**, seeks to estimate functions

$$f(\mathbf{x}) = (\mathbf{w} \cdot \mathbf{x}) + b, \quad \mathbf{w}, \mathbf{x} \in \mathbb{R}^N, b \in \mathbb{R}, \tag{1}$$

based on data

$$(\mathbf{x}_1, y_1), \ldots, (\mathbf{x}_\ell, y_\ell) \in \mathbb{R}^N \times \mathbb{R}, \tag{2}$$

by minimizing the regularized risk functional

$$\|\mathbf{w}\|^2 / 2 + C \cdot R_{emp}^\varepsilon, \tag{3}$$

where $C$ is a constant determining the trade-off between minimizing training errors and minimizing the model complexity term $\|\mathbf{w}\|^2$, and $R_{emp}^\varepsilon := \frac{1}{\ell} \sum_{i=1}^\ell |y_i - f(\mathbf{x}_i)|_\varepsilon$.

The parameter $\varepsilon$ can be useful if the desired accuracy of the approximation can be specified beforehand. In some cases, however, we just want the estimate to be as accurate as possible, without having to commit ourselves to a certain level of accuracy.

We present a modification of the $\varepsilon$-SVR algorithm which automatically minimizes $\varepsilon$, thus adjusting the accuracy level to the data at hand.

## 2  $\nu$-SV REGRESSION AND $\varepsilon$-SV REGRESSION

To estimate functions (1) from empirical data (2) we proceed as follows (Schölkopf et al., 1998a). At each point $\mathbf{x}_i$, we allow an error of $\varepsilon$. Everything above $\varepsilon$ is captured in slack variables $\xi_i^{(*)}$ ($(*)$ being a shorthand implying both the variables with and without asterisks), which are penalized in the objective function via a regularization constant $C$, chosen a priori (Vapnik, 1995). The tube size $\varepsilon$ is traded off against model complexity and slack variables via a constant $\nu \geq 0$:

$$\text{minimize} \quad \tau(\mathbf{w}, \boldsymbol{\xi}^{(*)}, \varepsilon) = \|\mathbf{w}\|^2/2 + C \cdot \left( \nu\varepsilon + \frac{1}{\ell} \sum_{i=1}^{\ell} (\xi_i + \xi_i^*) \right) \tag{4}$$

$$\text{subject to} \quad ((\mathbf{w} \cdot \mathbf{x}_i) + b) - y_i \;\leq\; \varepsilon + \xi_i \tag{5}$$

$$y_i - ((\mathbf{w} \cdot \mathbf{x}_i) + b) \;\leq\; \varepsilon + \xi_i^* \tag{6}$$

$$\xi_i^{(*)} \;\geq\; 0, \quad \varepsilon \;\geq\; 0. \tag{7}$$

Here and below, it is understood that $i = 1, \ldots, \ell$, and that bold face greek letters denote $\ell$-dimensional vectors of the corresponding variables. Introducing a Lagrangian with multipliers $\alpha_i^{(*)}, \eta_i^{(*)}, \beta \geq 0$, we obtain the the Wolfe dual problem. Moreover, as Boser et al. (1992), we substitute a kernel $k$ for the dot product, corresponding to a dot product in some feature space related to input space via a nonlinear map $\Phi$,

$$k(\mathbf{x}, \mathbf{y}) = (\Phi(\mathbf{x}) \cdot \Phi(\mathbf{y})). \tag{8}$$

This leads to the $\nu$-**SVR Optimization Problem**: for $\nu \geq 0, C > 0$,

$$\text{maximize} \;\; W(\boldsymbol{\alpha}^{(*)}) = \sum_{i=1}^{\ell} (\alpha_i^* - \alpha_i) y_i - \frac{1}{2} \sum_{i,j=1}^{\ell} (\alpha_i^* - \alpha_i)(\alpha_j^* - \alpha_j) k(\mathbf{x}_i, \mathbf{x}_j) \tag{9}$$

subject to

$$\sum_{i=1}^{\ell} (\alpha_i - \alpha_i^*) = 0, \;\; (10) \qquad 0 \leq \alpha_i^{(*)} \leq \frac{C}{\ell}, \;\; (11) \qquad \sum_{i=1}^{\ell} (\alpha_i + \alpha_i^*) \leq C \cdot \nu. \;\; (12)$$

The regression estimate can be shown to take the form

$$f(\mathbf{x}) = \sum_{i=1}^{\ell} (\alpha_i^* - \alpha_i) k(\mathbf{x}_i, \mathbf{x}) + b, \tag{13}$$

where $b$ (and $\varepsilon$) can be computed by taking into account that (5) and (6) (substitution of $\sum_j (\alpha_j^* - \alpha_j) k(\mathbf{x}_j, \mathbf{x})$ for $(\mathbf{w} \cdot \mathbf{x})$ is understood) become equalities with $\xi_i^{(*)} = 0$ for points with $0 < \alpha_i^{(*)} < C/\ell$, respectively, due to the Karush-Kuhn-Tucker conditions (cf. Vapnik, 1995). The latter moreover imply that in the kernel expansion (13), only those $\alpha_i^{(*)}$ will be nonzero that correspond to a constraint (5)/(6) which is precisely met. The respective patterns $\mathbf{x}_i$ are referred to as *Support Vectors*.

Before we give theoretical results explaining the significance of the parameter $\nu$, the following observation concerning $\varepsilon$ is helpful. If $\nu > 1$, then $\varepsilon = 0$, since it does not pay to increase $\varepsilon$ (cf. (4)). If $\nu \leq 1$, it can still happen that $\varepsilon = 0$, e.g. if the data are noise-free and can perfectly be interpolated with a low capacity model. The case $\varepsilon = 0$, however, is not what we are interested in; it corresponds to plain $L_1$ loss regression. Below, we will use the term **errors** to refer to training points lying outside of the tube, and the term **fraction** of errors/SVs to denote the relative numbers of errors/SVs, i.e. divided by $\ell$.

**Proposition 1** *Assume $\varepsilon > 0$. The following statements hold:*

    *(i) $\nu$ is an upper bound on the fraction of errors.*

    *(ii) $\nu$ is a lower bound on the fraction of SVs.*

*(iii) Suppose the data (2) were generated iid from a distribution $P(\mathbf{x}, y) = P(\mathbf{x})P(y|\mathbf{x})$ with $P(y|\mathbf{x})$ continuous. With probability 1, asymptotically, $\nu$ equals both the fraction of SVs and the fraction of errors.*

The first two statements of this proposition can be proven from the structure of the dual optimization problem, with (12) playing a crucial role. Presently, we instead give a graphical proof based on the primal problem (Fig. 1).

To understand the third statement, note that all errors are also SVs, but there can be SVs which are not errors: namely, if they lie exactly at the edge of the tube. Asymptotically, however, these SVs form a negligible fraction of the whole SV set, and the set of errors and the one of SVs essentially coincide. This is due to the fact that for a class of functions with well-behaved capacity (such as SV regression functions), and for a distribution satisfying the above continuity condition, the number of points that the tube edges $f \pm \varepsilon$ can pass through cannot asymptotically increase linearly with the sample size. Interestingly, the proof (Schölkopf et al., 1998a) uses a uniform convergence argument similar in spirit to those used in statistical learning theory.

Due to this proposition, $0 \leq \nu \leq 1$ can be used to control the number of errors (note that for $\nu \geq 1$, (11) implies (12), since $\alpha_i \cdot \alpha_i^* = 0$ for all $i$ (Vapnik, 1995)). Moreover, since the constraint (10) implies that (12) is equivalent to $\sum_i \alpha_i^{(*)} \leq C\nu/2$, we conclude that Proposition 1 actually holds for the upper and the lower edge of the tube separately, with $\nu/2$ each. As an aside, note that by the same argument, the number of SVs at the two edges of the standard $\varepsilon$-SVR tube asymptotically agree.

Moreover, note that this bears on the *robustness* of $\nu$-SVR. At first glance, SVR seems all but robust: using the $\varepsilon$-insensitive loss function, only the patterns *outside* of the $\varepsilon$-tube contribute to the empirical risk term, whereas the patterns *closest* to the estimated regression have zero loss. This, however, does not mean that it is only the outliers that determine the regression. In fact, the contrary is the case: one can show that local movements of target values $y_i$ of points $\mathbf{x}_i$ outside the tube do not influence the regression (Schölkopf et al., 1998c). Hence, $\nu$-SVR is a generalization of an estimator for the mean of a random variable which throws away the largest and smallest examples (a fraction of at most $\nu/2$ of either category), and estimates the mean by taking the average of the two extremal ones of the remaining examples. This is close in spirit to robust estimators like the *trimmed mean*.

Let us briefly discuss how the new algorithm relates to $\varepsilon$-SVR (Vapnik, 1995). By rewriting (3) as a constrained optimization problem, and deriving a dual much like we did for $\nu$-SVR,

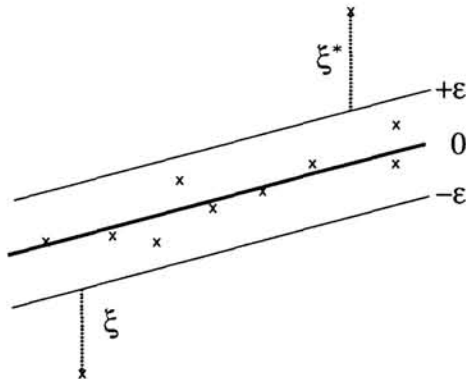

Figure 1: Graphical depiction of the $\nu$-trick. Imagine increasing $\varepsilon$, starting from 0. The first term in $\nu\varepsilon + \frac{1}{\ell}\sum_{i=1}^{\ell}(\xi_i + \xi_i^*)$ (cf. (4)) will increase proportionally to $\nu$, while the second term will decrease proportionally to the fraction of points outside of the tube. Hence, $\varepsilon$ will grow as long as the latter fraction is larger than $\nu$. At the optimum, it therefore must be $\leq \nu$ (Proposition 1, (i)). Next, imagine decreasing $\varepsilon$, starting from some large value. Again, the change in the first term is proportional to $\nu$, but this time, the change in the second term is proportional to the fraction of SVs (even points *on* the edge of the tube will contribute). Hence, $\varepsilon$ will shrink as long as the fraction of SVs is smaller than $\nu$, eventually leading to Proposition 1, (ii).

one arrives at the following quadratic program: maximize

$$W(\boldsymbol{\alpha}, \boldsymbol{\alpha}^*) = -\varepsilon \sum_{i=1}^{\ell} (\alpha_i^* + \alpha_i) + \sum_{i=1}^{\ell} (\alpha_i^* - \alpha_i)y_i - \frac{1}{2} \sum_{i,j=1}^{\ell} (\alpha_i^* - \alpha_i)(\alpha_j^* - \alpha_j)k(\mathbf{x}_i, \mathbf{x}_j) \quad (14)$$

subject to (10) and (11). Compared to (9), we have an additional term $-\varepsilon \sum_{i=1}^{\ell}(\alpha_i^* + \alpha_i)$, which makes it plausible that the constraint (12) is not needed.

In the following sense, $\nu$-SVR includes $\varepsilon$-SVR. Note that in the general case, using kernels, $\bar{\mathbf{w}}$ is a vector in feature space.

**Proposition 2** *If $\nu$-SVR leads to the solution $\bar\varepsilon, \bar{\mathbf{w}}, \bar{b}$, then $\varepsilon$-SVR with $\varepsilon$ set a priori to $\bar\varepsilon$, and the same value of $C$, has the solution $\bar{\mathbf{w}}, \bar{b}$.*

**Proof** If we minimize (4), then fix $\varepsilon$ and minimize only over the remaining variables, the solution does not change. ∎

## 3 PARAMETRIC INSENSITIVITY MODELS

We generalized $\varepsilon$-SVR by considering the tube as not given but instead estimated it as a model parameter. What we have so far retained is the assumption that the $\varepsilon$-insensitive zone has a tube (or slab) shape. We now go one step further and use parametric models of arbitrary shape. Let $\{\zeta_q^{(*)}\}$ (here and below, $q = 1, \ldots, p$ is understood) be a set of $2p$ positive functions on $\mathbb{R}^N$. Consider the following quadratic program: for given $\nu_1^{(*)}, \ldots, \nu_p^{(*)} \geq 0$, minimize

$$\tau(\mathbf{w}, \boldsymbol{\xi}^{(*)}, \boldsymbol{\varepsilon}^{(*)}) = \|\mathbf{w}\|^2/2 + C \cdot \left( \sum_{q=1}^{p} (\nu_q \varepsilon_q + \nu_q^* \varepsilon_q^*) + \frac{1}{\ell} \sum_{i=1}^{\ell} (\xi_i + \xi_i^*) \right) \quad (15)$$

subject to
$$((\mathbf{w} \cdot \mathbf{x}_i) + b) - y_i \leq \sum_q \varepsilon_q \zeta_q(\mathbf{x}_i) + \xi_i \quad (16)$$

$$y_i - ((\mathbf{w} \cdot \mathbf{x}_i) + b) \leq \sum_q \varepsilon_q^* \zeta_q^*(\mathbf{x}_i) + \xi_i^* \quad (17)$$

$$\xi_i^{(*)} \geq 0, \quad \varepsilon_q^{(*)} \geq 0. \quad (18)$$

A calculation analogous to Sec. 2 shows that the Wolfe dual consists of maximizing (9) subject to (10), (11), and, instead of (12), the modified constraints $\sum_{i=1}^{\ell} \alpha_i^{(*)} \zeta_q^{(*)}(\mathbf{x}_i) \leq C \cdot \nu_q^{(*)}$. In the experiments in Sec. 4, we use a simplified version of this optimization problem, where we drop the term $\nu_q^* \varepsilon_q^*$ from the objective function (15), and use $\varepsilon_q$ and $\zeta_q$ in (17). By this, we render the problem symmetric with respect to the two edges of the tube. In addition, we use $p = 1$. This leads to the same Wolfe dual, except for the last constraint, which becomes (cf. (12))

$$\sum_{i=1}^{\ell} (\alpha_i + \alpha_i^*)\zeta(\mathbf{x}_i) \leq C \cdot \nu. \quad (19)$$

The advantage of this setting is that since the same $\nu$ is used for both sides of the tube, the computation of $\varepsilon, b$ is straightforward: for instance, by solving a linear system, using two conditions as those described following (13). Otherwise, general statements are harder to make: the linear system can have a zero determinant, depending on whether the functions $\zeta_p^{(*)}$, evaluated on the $\mathbf{x}_i$ with $0 < \alpha_i^{(*)} < C/\ell$, are linearly dependent. The latter occurs, for instance, if we use constant functions $\zeta^{(*)} \equiv 1$. In this case, it is pointless to use two different values $\nu, \nu^*$; for, the constraint (10) then implies that *both* sums $\sum_{i=1}^{\ell} \alpha_i^{(*)}$ will be bounded by $C \cdot \min\{\nu, \nu^*\}$. We conclude this section by giving, without proof, a generalization of Proposition 1, (iii), to the optimization problem with constraint (19):

**Proposition 3** *Assume $\varepsilon > 0$. Suppose the data (2) were generated iid from a distribution $P(\mathbf{x}, y) = P(\mathbf{x})P(y|\mathbf{x})$ with $P(y|\mathbf{x})$ continuous. With probability 1, asymptotically, the fractions of SVs and errors equal $\nu \cdot (\int \zeta(\mathbf{x})\, d\tilde{P}(\mathbf{x}))^{-1}$, where $\tilde{P}$ is the asymptotic distribution of SVs over $\mathbf{x}$.*

## 4    EXPERIMENTS AND DISCUSSION

In the experiments, we used the optimizer LOQO (http://www.princeton.edu/rvdb/). This has the serendipitous advantage that the primal variables $b$ and $\varepsilon$ can be recovered as the dual variables of the Wolfe dual (9) (i.e. the double dual variables) fed into the optimizer.

In Fig. 2, the task was to estimate a regression of a noisy sinc function, given $\ell$ examples $(x_i, y_i)$, with $x_i$ drawn uniformly from $[-3, 3]$, and $y_i = \sin(\pi x_i)/(\pi x_i) + v_i$, with $v_i$ drawn from a Gaussian with zero mean and variance $\sigma^2$. We used the default parameters $\ell = 50$, $C = 100$, $\sigma = 0.2$, and the RBF kernel $k(x, x') = \exp(-|x - x'|^2)$.

Figure 3 gives an illustration of how one can make use of parametric insensitivity models as proposed in Sec. 3. Using the proper model, the estimate gets much better. In the parametric case, we used $\nu = 0.1$ and $\zeta(x) = \sin^2((2\pi/3)x)$, which, due to $\int \zeta(x)\, dP(x) = 1/2$, corresponds to our standard choice $\nu = 0.2$ in $\nu$-SVR (cf. Proposition 3). The experimental findings are consistent with the asymptotics predicted theoretically even if we assume a uniform distribution of SVs: for $\ell = 200$, we got 0.24 and 0.19 for the fraction of SVs and errors, respectively.

This method allows the incorporation of prior knowledge into the loss function. Although this approach at first glance seems fundamentally different from incorporating prior knowledge directly into the kernel (Schölkopf et al., 1998b), from the point of view of statistical

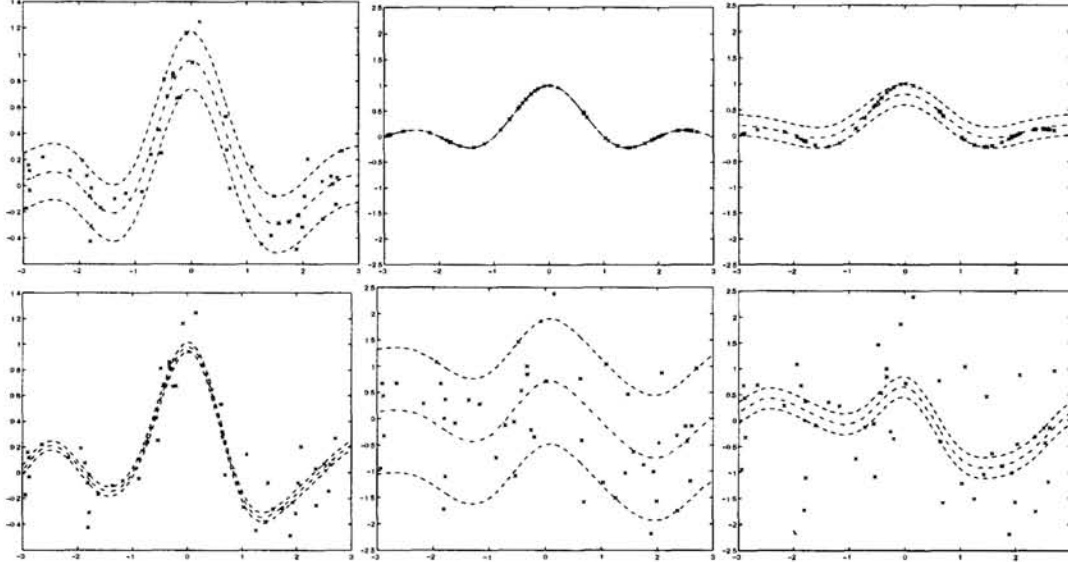

Figure 2: *Left:* $\nu$-SV regression with $\nu = 0.2$ (top) and $\nu = 0.8$ (bottom). The larger $\nu$ allows more points to lie outside the tube (see Sec. 2). The algorithm automatically adjusts $\varepsilon$ to 0.22 (top) and 0.04 (bottom). Shown are the sinc function (dotted), the regression $f$ and the tube $f \pm \varepsilon$. *Middle:* $\nu$-SV regression on data with noise $\sigma = 0$ (top) and $\sigma = 1$ (bottom). In both cases, $\nu = 0.2$. The tube width automatically adjusts to the noise (top: $\varepsilon = 0$, bottom: $\varepsilon = 1.19$). *Right:* $\varepsilon$-SV regression (Vapnik, 1995) on data with noise $\sigma = 0$ (top) and $\sigma = 1$ (bottom). In both cases, $\varepsilon = 0.2$ — this choice, which has to be specified a priori, is ideal for neither case: in the top figure, the regression estimate is biased; in the bottom figure, $\varepsilon$ does not match the external noise (cf. Smola et al., 1998).

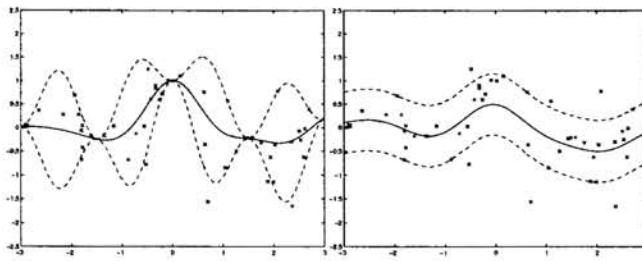

Figure 3: Toy example, using prior knowledge about an $x$-dependence of the noise. Additive noise ($\sigma = 1$) was multiplied by $\sin^2((2\pi/3)x)$. *Left:* the *same* function was used as $\zeta$ as a parametric insensitivity tube (Sec. 3). *Right:* $\nu$-SVR with standard tube.

Table 1: Results for the Boston housing benchmark; *top:* $\nu$-SVR, *bottom:* $\varepsilon$-SVR. MSE: Mean squared errors, STD: standard deviations thereof (100 trials), Errors: fraction of training points outside the tube, SVs: fraction of training points which are SVs.

| $\nu$ | 0.1 | 0.2 | 0.3 | 0.4 | 0.5 | 0.6 | 0.7 | 0.8 | 0.9 | 1.0 |
|---|---|---|---|---|---|---|---|---|---|---|
| automatic $\varepsilon$ | 2.6 | 1.7 | 1.2 | 0.8 | 0.6 | 0.3 | 0.0 | 0.0 | 0.0 | 0.0 |
| MSE | 9.4 | 8.7 | 9.3 | 9.5 | 10.0 | 10.6 | 11.3 | 11.3 | 11.3 | 11.3 |
| STD | 6.4 | 6.8 | 7.6 | 7.9 | 8.4 | 9.0 | 9.6 | 9.5 | 9.5 | 9.5 |
| Errors | 0.0 | 0.1 | 0.2 | 0.2 | 0.3 | 0.4 | 0.5 | 0.5 | 0.5 | 0.5 |
| SVs | 0.3 | 0.4 | 0.6 | 0.7 | 0.8 | 0.9 | 1.0 | 1.0 | 1.0 | 1.0 |

| $\varepsilon$ | 0 | 1 | 2 | 3 | 4 | 5 | 6 | 7 | 8 | 9 | 10 |
|---|---|---|---|---|---|---|---|---|---|---|---|
| MSE | 11.3 | 9.5 | 8.8 | 9.7 | 11.2 | 13.1 | 15.6 | 18.2 | 22.1 | 27.0 | 34.3 |
| STD | 9.5 | 7.7 | 6.8 | 6.2 | 6.3 | 6.0 | 6.1 | 6.2 | 6.6 | 7.3 | 8.4 |
| Errors | 0.5 | 0.2 | 0.1 | 0.0 | 0.0 | 0.0 | 0.0 | 0.0 | 0.0 | 0.0 | 0.0 |
| SVs | 1.0 | 0.6 | 0.4 | 0.3 | 0.2 | 0.1 | 0.1 | 0.1 | 0.1 | 0.1 | 0.1 |

learning theory the two approaches are closely related: in both cases, the structure of the loss-function-induced class of functions (which is the object of interest for generalization error bounds) is customized; in the first case, by changing the loss function, in the second case, by changing the class of functions that the estimate is taken from.

Empirical studies using $\varepsilon$-SVR have reported excellent performance on the widely used **Boston housing regression benchmark** set (Stitson et al., 1999). Due to Proposition 2, the only difference between $\nu$-SVR and standard $\varepsilon$-SVR lies in the fact that different parameters, $\varepsilon$ vs. $\nu$, have to be specified a priori. Consequently, we are in this experiment only interested in these parameters and simply adjusted $C$ and the width $2\sigma^2$ in $k(\mathbf{x}, \mathbf{y}) = \exp(-\|\mathbf{x} - \mathbf{y}\|^2/(2\sigma^2))$ as Schölkopf et al. (1997): we used $2\sigma^2 = 0.3 \cdot N$, where $N = 13$ is the input dimensionality, and $C/\ell = 10 \cdot 50$ (i.e. the original value of 10 was corrected since in the present case, the maximal $y$-value is 50). We performed 100 runs, where each time the overall set of 506 examples was randomly split into a training set of $\ell = 481$ examples and a test set of 25 examples. Table 1 shows that in a wide range of $\nu$ (note that only $0 \leq \nu \leq 1$ makes sense), we obtained performances which are close to the best performances that can be achieved by selecting $\varepsilon$ a priori by looking at the test set. Finally, note that although we did not use validation techniques to select the optimal values for $C$ and $2\sigma^2$, we obtained performance which are state of the art (Stitson et al. (1999) report an MSE of 7.6 for $\varepsilon$-SVR using ANOVA kernels, and 11.7 for Bagging trees). Table 1 moreover shows that $\nu$ can be used to control the fraction of SVs/errors.

**Discussion.** The theoretical and experimental analysis suggest that $\nu$ provides a way to control an upper bound on the number of training errors which is tighter than the one used in the soft margin hyperplane (Vapnik, 1995). In many cases, this makes it a parameter which is more convenient than the one in $\varepsilon$-SVR. Asymptotically, it directly controls the

number of Support Vectors, and the latter can be used to give a leave-one-out generalization bound (Vapnik, 1995). In addition, $\nu$ characterizes the compression ratio: it suffices to train the algorithm only on the SVs, leading to the same solution (Schölkopf et al., 1995). In $\varepsilon$-SVR, the tube width $\varepsilon$ must be specified a priori; in $\nu$-SVR, which generalizes the idea of the *trimmed mean*, it is computed automatically. Desirable properties of $\varepsilon$-SVR, including the formulation as a definite quadratic program, and the sparse SV representation of the solution, are retained. We are optimistic that in many applications, $\nu$-SVR will be more robust than $\varepsilon$-SVR. Among these should be the reduced set algorithm of Osuna and Girosi (1999), which approximates the SV pattern recognition decision surface by $\varepsilon$-SVR. Here, $\nu$ should give a direct handle on the desired speed-up.

One of the immediate questions that a $\nu$-approach to SV regression raises is whether a similar algorithm is possible for the case of pattern recognition. This question has recently been answered to the affirmative (Schölkopf et al., 1998c). Since the pattern recognition algorithm (Vapnik, 1995) does not use $\varepsilon$, the only parameter that we can dispose of by using $\nu$ is the regularization constant $C$. This leads to a dual optimization problem with a homogeneous quadratic form, and $\nu$ lower bounding the sum of the Lagrange multipliers. Whether we could have abolished $C$ in the regression case, too, is an open problem.

**Acknowledgement**    This work was supported by the ARC and the DFG (# Ja 379/71).

# References

B. E. Boser, I. M. Guyon, and V. N. Vapnik. A training algorithm for optimal margin classifiers. In D. Haussler, editor, *Proceedings of the 5th Annual ACM Workshop on Computational Learning Theory*, pages 144–152, Pittsburgh, PA, 1992. ACM Press.

E. Osuna and F. Girosi. Reducing run-time complexity in support vector machines. In B. Schölkopf, C. Burges, and A. Smola, editors, *Advances in Kernel Methods — Support Vector Learning*, pages 271 – 283. MIT Press, Cambridge, MA, 1999.

B. Schölkopf, C. Burges, and V. Vapnik. Extracting support data for a given task. In U. M. Fayyad and R. Uthurusamy, editors, *Proceedings, First International Conference on Knowledge Discovery & Data Mining*. AAAI Press, Menlo Park, CA, 1995.

B. Schölkopf, P. Bartlett, A. Smola, and R. Williamson. Support vector regression with automatic accuracy control. In L. Niklasson, M. Bodén, and T. Ziemke, editors, *Proceedings of the 8th International Conference on Artificial Neural Networks*, Perspectives in Neural Computing, pages 111 – 116, Berlin, 1998a. Springer Verlag.

B. Schölkopf, P. Simard, A. Smola, and V. Vapnik. Prior knowledge in support vector kernels. In M. Jordan, M. Kearns, and S. Solla, editors, *Advances in Neural Information Processing Systems 10*, pages 640 – 646, Cambridge, MA, 1998b. MIT Press.

B. Schölkopf, A. Smola, R. Williamson, and P. Bartlett. New support vector algorithms. 1998c. NeuroColt2-TR 1998-031; cf. http:/www.neurocolt.com

B. Schölkopf, K. Sung, C. Burges, F. Girosi, P. Niyogi, T. Poggio, and V. Vapnik. Comparing support vector machines with gaussian kernels to radial basis function classifiers. *IEEE Trans. Sign. Processing*, 45:2758 – 2765, 1997.

A. Smola, N. Murata, B. Schölkopf, and K.-R. Müller. Asymptotically optimal choice of $\varepsilon$-loss for support vector machines. In L. Niklasson, M. Bodén, and T. Ziemke, editors, *Proceedings of the 8th International Conference on Artificial Neural Networks*, Perspectives in Neural Computing, pages 105 – 110, Berlin, 1998. Springer Verlag.

M. Stitson, A. Gammerman, V. Vapnik, V. Vovk, C. Watkins, and J. Weston. Support vector regression with ANOVA decomposition kernels. In B. Schölkopf, C. Burges, and A. Smola, editors, *Advances in Kernel Methods — Support Vector Learning*, pages 285 – 291. MIT Press, Cambridge, MA, 1999.

V. Vapnik. *The Nature of Statistical Learning Theory*. Springer Verlag, New York, 1995.
